# High-Performance Job-Shop Scheduling With A Time-Delay TD($\lambda$) Network

**Wei Zhang and Thomas G. Dietterich**
Department of Computer Science
Oregon State University
Corvallis, Oregon 97331-3202
{zhangw, tgd}@research.cs.orst.edu

## Abstract

Job-shop scheduling is an important task for manufacturing industries. We are interested in the particular task of scheduling payload processing for NASA's space shuttle program. This paper summarizes our previous work on formulating this task for solution by the reinforcement learning algorithm $TD(\lambda)$. A shortcoming of this previous work was its reliance on hand-engineered input features. This paper shows how to extend the time-delay neural network (TDNN) architecture to apply it to irregular-length schedules. Experimental tests show that this TDNN-$TD(\lambda)$ network can match the performance of our previous hand-engineered system. The tests also show that both neural network approaches significantly outperform the best previous (non-learning) solution to this problem in terms of the quality of the resulting schedules and the number of search steps required to construct them.

## 1 Introduction

In Tesauro's 1992 landmark work on TD-gammon, he showed that the temporal difference algorithm $TD(\lambda)$ [Sutton, 1988] can learn an excellent evaluation function for the game of backgammon. This is the most successful application of reinforcement learning to date. The goal of our research is to determine whether this success can be duplicated in an application of industrial importance: Job-shop scheduling.

We are interested in a particular scheduling problem: space shuttle payload processing for NASA. The goal is to schedule a set of tasks to satisfy a set of temporal and resource constraints while also seeking to minimize the total duration (makespan) of the schedule. The best existing method for this task is an iterative repair scheduler that combines heuristics with simulated annealing [Zweben et al., 1994]. In [Zhang

and Dietterich, 1995], we report initial results showing that a neural network-based $TD(\lambda)$ scheduler can out-perform this iterative repair algorithm.

To obtain those results, we hand-engineered a set of input features. An advantage of neural network algorithms, however, is that they can often learn good "features" (i.e., hidden units) from more primitive, raw features. The work described in this paper shows how to apply the time-delay neural network architecture [Lang *et al.*, 1990, LeCun *et al.*, 1989] to this task to learn from raw features and thereby eliminate hand-engineering.

In the following sections, we first describe the scheduling task and show how this task can be formulated for TD($\lambda$). We then discuss the problem of schedule representation and our network architecture. Following this, we present experiments on a set of simulated problems and discuss the results. These results show that the time-delay network using low level features can not only match the performance of the hand-engineered features—it can actually perform slightly better.

## 2   The NASA Domain and TD($\lambda$) for the Task

The NASA space shuttle payload processing (SSPP) domain requires scheduling the tasks that must be performed to install and test the payloads that are placed in the cargo bay of the space shuttle. In job-shop scheduling terminology, each shuttle mission is a job, which has a fixed launch date. Each job consists of a partially-ordered set of tasks that must be performed. Most of these tasks are "pre-tasks" that must be performed prior to launch, but some are "post-tasks" that take place after the shuttle has landed. Each task has a duration and a list of resource requirements. The resources are grouped into resource pools. For each task and each type of resource, the required amount of the resource must be obtained from a single resource pool. A complete schedule must specify the start time of each task and the resource pool by which each resource requirement of each task is satisfied. A key goal of the scheduling system is to minimize the total duration of the schedule. This is much more challenging than simply finding a *feasible* schedule.

Zweben et al. [1994] developed the following iterative repair method for solving this scheduling problem. First, a critical path schedule is constructed by working backward and forward from the launch and landing dates; resource constraints are ignored. This critical-path schedule serves as the starting state for a state-space search. In each state of this problem space, there are two possible operators that can be applied. The REASSIGN-POOL operator changes the pool assignment for one of the resource requirements of a task. It is only applied when the pool reassignment would allow the resource requirement to be successfully satisfied. The MOVE operator moves a task to a different time and then reschedules all of the temporal dependents of the task using the critical path method (leaving the resource pool assignments of the dependents unchanged). The MOVE operator is only applied to move a task to the first earlier or the first later time at which the violated resource requirement can be satisfied. The iterative repair method uses a combination of three heuristics to choose a task to repair. It prefers to move the task that (a) requires an amount of resource nearly equal to the amount that is over allocated, (b) has few temporal dependents, and (c) needs to be moved only a short distance to satisfy the resource request. The overall control structure of the algorithm applies simulated annealing to minimize a designated cost function. The search continues until a schedule is found that has no constraint violations.

To view the scheduling problem as a reinforcement learning problem, we must describe the problem space and the reinforcement function. We employ the same

problem space as Zweben et al. The starting state $s_0$ is the critical path schedule as discussed above. We define the reinforcement function $R(s)$ to give a reinforcement of $-0.001$ for each schedule $s$ that still contains constraint violations. This assesses a small penalty for each scheduling action, and it is intended to encourage reinforcement learning to prefer actions that *quickly* find a good schedule. For any schedule $s$ that is free of violations, the reinforcement is the negative of the *resource dilation factor*, $-RDF(s, s_0)$. The RDF attempts to provide a scale-independent measure of the length of the schedule, and this final reinforcement is intended to encourage reinforcement learning to find short final schedules.

The *RDF* is defined as follows. Let *capacity(i)* be the capacity of resource type $i$—that is, the combined capacity of all resource pools of resource type $i$. At each time $t$ in the schedule, let $u(i, t)$ be the current utilization of resources of type $i$. We define the *resource utilization index* $RUI(i, t)$ for resource type $i$ at time $t$ to be $RUI(i, t) = \max\left\{1, \frac{u(i,t)}{capacity(i)}\right\}$. If the resource is not over-allocated, $RUI(i, t)$ is 1; otherwise it is the fraction of overallocation. The *total resource utilization index* ($TRUI$) for a schedule of length $l$ is the sum of the resource utilization index taken over all $n$ resources and all $l$ times: $TRUI = \sum_{i=1}^{n} \sum_{t=1}^{l} RUI(i, t)$. Given these definitions, the resource dilation factor is defined as $RDF(s, s_0) = \frac{TRUI(s)}{TRUI(s_0)}$.

Now that we have specified how to view repair-based scheduling as a reinforcement learning problem, we turn our attention to the learning algorithm. Suppose at a given point in the learning process we have developed policy $\pi$, which says that in state $s$ the action to select is $a = \pi(s)$. We can define an associated function $f_\pi$, called the *value function*, such that $f_\pi(s)$ tells the cumulative reward that we will receive if we follow policy $\pi$ from state $s$ onward. Formally, $f_\pi(s) = \sum_{j=0}^{N} R(s_{j+1})$, where $N$ is the number of steps until a conflict-free schedule is found.

As in most reinforcement learning work, we will attempt to learn the value function of the optimal policy $\pi^*$, denoted $f^* = f_{\pi^*}$, rather than directly learning $\pi^*$. Once we have learned this optimal value function, we can transform it into the optimal policy via a simple one-step lookahead search. To learn the value function, we apply $TD(\lambda)$ as a form of value iteration. $TD(\lambda)$ is applied online to the sequences of states and reinforcements that result from choosing actions according to the current estimated value function, $\hat{f}$. At each state $s$ during learning, we conduct a one-step lookahead search using the current estimated value function $\hat{f}$ to evaluate the states resulting from applying each possible operator. We then select the action that maximizes the predicted value of the resulting state $s'$. After applying this action and receiving the reward, we update our estimate of $\hat{f}$ to reflect the difference between the value of $\hat{f}(s)$ and the more informed value $R(s') + \hat{f}(s')$.

## 3  Schedule Representation and Network Design

The main challenge for designing a schedule representation is that virtually all methods for learning evaluation functions can only be applied to fixed-length vectors of features. However, the length of schedules varies depending on the number of tasks and the complexity of their temporal and resource constraints. In our previous work, we hand-engineered a fixed set of features that summarized the structure of the schedule. We included such features as the RDF of the current schedule, the mean and standard deviation of the unused resource capacity of each resource pool (negative if the pool is over-allocated), the mean and standard deviation of the slack times (idle times between temporal dependents), and so on. However,

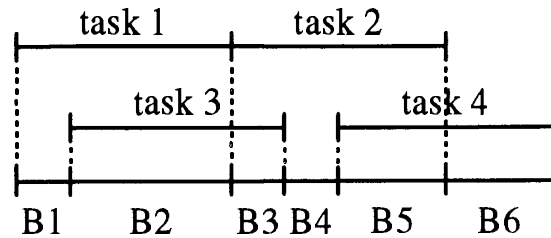

Figure 1: The definition of "blocks".

hand-engineering increases the cost of creating a new application and reduces the autonomy of the learning system. Therefore, we wish to develop a method that can automatically learn good input features.

The time-delay neural network [Lang et al., 1990] has proved to be very effective in learning good position-independent features in visual- and speech-recognition tasks. In speech recognition, it is applied to convert an input sequence of speech frames into a "hidden sequence" of extracted features. A classifier then walks along this hidden sequence and classifies each position in the sequence. The mapping from a particular position in the input sequence to a particular position in the hidden sequence is performed by a "kernel" neural network, which is scanned along the input sequence. It examines a sliding window of adjacent positions in the sequence. The kernel network has a single set of weights that are applied at all positions in the sequence, although each position may have its own bias weight.

To apply this architecture to scheduling, we must solve two problems. First, we must define what a "position" means in the schedule. Second, we must decide how to use the "hidden sequence" of computed features.

To define "positions", we subdivide the schedule into a sequence of "blocks." Each block is a maximal time interval in which the current tasks and resource assignments do not change (see Figure 1). For each block, we can compute primitive features such as the number of resource units available in each pool, and whether each pool is over-allocated. We also say that a task is "inside" a block if that task starts at the beginning of the block. Given this definition, we can compute additional primitive features of the tasks inside the block: minimum and average slack time, number of dependents, and number of tasks inside the block. If there are multiple tasks inside a block, we compute the average values of these features. A total of 12 primitive features are computed for each block.

We then define five kernels, each of which examines a single "current block" and computes a hidden "feature" from this block. These five kernels are scanned along the entire schedule, and they create a derived sequence containing five "hidden features" for each block. How should this derived sequence be processed to compute the estimated value function?

One approach would be to scan a network along the hidden sequence and take the maximum value output by that network. However, because our goal is to estimate the RDF of the final schedule, it seems wiser to view the kernels as learning to recognize "bad conflicts" and "opportunities". By summing (or averaging) the "hidden features", the network can effectively count the number of opportunities and conflicts and estimate how much more the RDF will change before the conflicts are eliminated. Hence, we take the following approach.

The sequence of "hidden features" is split into thirds. Within each third, five

features are computed by finding the mean value of each of the five hidden features over the blocks in this third. This gives a total of 15 features. These features, along with two other "global" features—the RDF of the current schedule and the resource utilization of the starting schedule—are input to a network having 40 hidden units. The output of that network is taken as the estimated value for the current schedule.

To recap, the network has three hidden layers named H1, H2, and H3. Layer H1 has 5 kernels (each with 12 weights). There are 15 biases, one for each feature in each third of the schedule. So H1 has a total of 75 parameters. H2 has 17 units; 15 of these are hidden units (3 sets of 5) averaged from H1 and 2 of these are the two global input features. H2 has no adjustable parameters. H3 has 40 hidden units fully connected to H2, for a total of 720 parameters. The output layer has 8 units fully connected to H3 and encoding the predicted RDF using the technique of overlapping gaussian ranges [Pomerleau, 1991]. The output layer has $328 (= 8 \cdot (40 + 1))$ parameters. Therefore, this net has a total of 1123 parameters. All units in H1, H3, and the output layer use sigmoidal transfer functions.

# 4 Experiments

We constructed an artificial problem set based on specifications for the NASA SSPP problem. Space constraints do not permit a complete description of the problems or the training procedure (see [Zhang and Dietterich, 1995] for full details). 100 scheduling problems were generated. These were subdivided into 50 problems for final testing, 20 problems for validation testing, and three training sets of 10 problems each. An ensemble of 6 networks was constructed by training a separate network for each of these three training sets and for $\lambda = 0.2$ and $\lambda = 0.7$. Training was monitored by testing on the validation set every 100 epochs. Training was halted when the validation test showed no further change. For each of the 6 networks, the final set of weights and the set of weights giving the best validation score were retained for a total of 12 networks.

Figure 2 compares the test set performance of six different scheduling configurations. G1TDN is the mean test set RDF of the best single TDNN $TD(\lambda)$ network (as determined by validation set performance). G12TDN is the mean RDF of all 12 learned networks. Analogously, G1N is the best single network trained using our hand-engineering features and G12N is the mean RDF of 12 hand-engineered networks. IR-V and IR-RDF are Zweben's iterative repair algorithms using the number of violations and the RDF (respectively) as the error function to be minimized via simulated annealing. From this we can see that G1TDN produces schedules with better average RDF than any of the other methods. In particular the F test shows that it is significantly better than all other algorithms except G12TDN.

Figure 3 compares the mean number of repairs required by each algorithm to find a solution. For the IR algorithms only repairs accepted by simulated annealing were counted. This shows that the neural network methods have learned very good evaluation functions—they are able to find a good solution much more directly than the simulated annealing methods. According to the F tests, the networks with engineered features are slightly better than the TDNN networks, but all of the networks are significantly better than simulated annealing.

A shortcoming of these figures, however, is that they only record the mean results of *single runs*. Better results are typically obtained from simulated annealing if the algorithm is run many times and the best solution retained. Figures 4 and 5 show the mean RDF of the best schedule as a function of the number of schedule repairs and CPU time expended. IR-V and IR-RDF were each run 50 times; G12N and

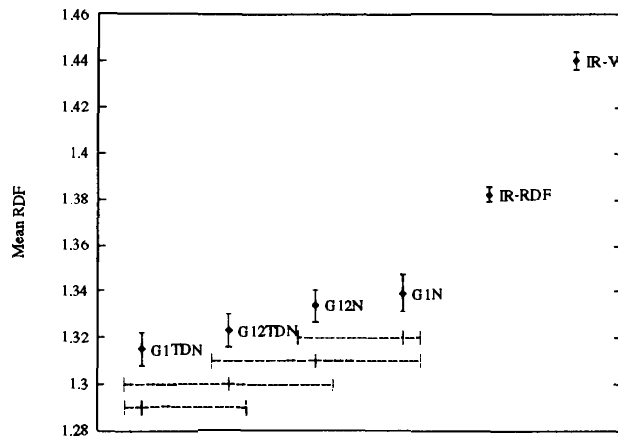
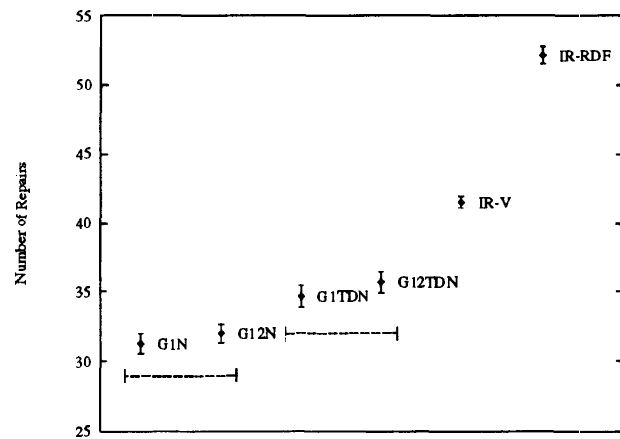

Figure 2: Mean RDF values. Vertical error bars show 95% confidence intervals. Horizontal bars group together algorithms that cannot be distinguished based on ANOVA $F$-tests at $p < 0.05$.

Figure 3: Mean number of repairs. Vertical error bars show 95% confidence intervals. Horizontal bars group together algorithms that cannot be distinguished based on ANOVA $F$-tests at $p < 0.05$.

G12TDN show the results of running each of the 12 networks once on each test problem. G1N and G1TDN show the results of running the best single net 10 times on each test problem. Because evaluation function ties are broken randomly, these 10 repetitions usually generate different schedules. Each time a network finds an improved solution to a problem, a point is plotted on the graph.

The graphs show that the learned networks clearly out-perform Zweben's IR algorithms. Figure 4 shows that the networks perform many fewer repairs to find schedules of the same quality as the IR algorithms. Note that the horizontal axis is plotted on a log scale—the networks maintain a constant *factor* advantage over IR. For a schedule of a year's duration, this improvement would translate into several days (and hundreds of thousands of dollars) saved.

Ultimately, G12TDN does slightly better than G12N. After 12 iterations and 21,676 repairs, G12TDN produced solutions with an average RDF of 1.196. By comparison, G12N performs 19,406 repairs and produces an average RDF of 1.202.

Figure 5 illustrates a problem with the neural network approach: the networks spend more CPU time selecting each repair. This reduces the differences between the methods. G12N exhibits the best tradeoff between CPU time and schedule quality, although G12TDN attains the best final schedule quality.

The major CPU cost of G12TDN is the cost of breaking the schedule into blocks and convolving the kernel networks with the blocks. There are many opportunities to make this more efficient by taking advantage of the fact that each repair changes only parts of the schedule, and therefore, only parts of the neural network calculation need to be updated.

## 5   Conclusions

This paper has shown how to apply temporal difference learning to job shop scheduling problems by formulating them as iterative repair problem spaces. The paper has also presented a modification of the TDNN architecture appropriate for schedul-

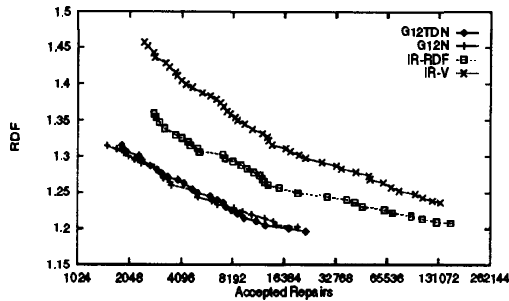
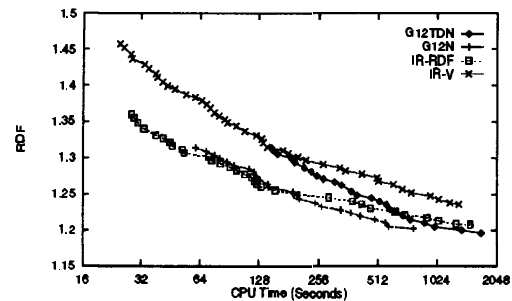

Figure 4: Comparison of Accepted Schedule Repairs

Figure 5: Comparison in CPU time

ing problems. The combined TDNN-$TD(\lambda)$ architecture can learn very powerful search heuristics that significantly out-perform all previous algorithms in terms of the quality of the resulting schedules. The TDNN architecture achieves this high performance with much less "feature-engineering" than our previous neural network approach. This demonstrates once again the superior ability of neural networks to learn useful higher-level features from raw input features.

Both of our neural-net-based methods demonstrate that the impressive performance of Tesauro's TD-gammon system can carry over to an important industrial application. Temporal difference learning is able to learn a very effective evaluation function for job shop scheduling. Using this function, a scheduler can find better schedules and find them in fewer search steps than the best previous methods.

# Acknowledgments

The authors thank Rich Sutton and Monte Zweben for several helpful discussions. The authors gratefully acknowledge the support of NASA grant NAG 2-630 from NASA Ames Research Center. Additional support was provided by NSF grants CDA-9216172 and IRI-9204129.

# References

[Lang et al., 1990] K. J. Lang, A. H. Waibel, and G. E. Hinton. A time-delay neural network architecture for isolated word recognition. *Neural Networks*, 3:33–43, 1990.

[LeCun et al., 1989] Y. LeCun, B. Boser, J. S. Deniker, and D. Henderson et al. Backpropagation applied to handwritten zip code recognition. *Neural Computation*, 1:541–551, 1989.

[Pomerleau, 1991] D. A. Pomerleau. Efficient training of artificial neural networks for autonomous navigation. *Neural Computation*, 3(1):88–97, 1991.

[Sutton, 1988] R. S. Sutton. Learning to predict by the methods of temporal differences. *Machine Learning*, 3(1):9–44, August 1988.

[Tesauro, 1992] G. J. Tesauro. Practical issues in temporal difference learning. *Machine Learning*, 8(3/4):257–277, 1992.

[Zhang and Dietterich, 1995] W. Zhang and T. Dietterich. A reinforcement learning approach to job-shop scheduling. In *IJCAI-95*, pages 1114–1120, 1995.

[Zweben et al., 1994] M. Zweben, B. Daun, and M. Deal. Scheduling and rescheduling with iterative repair. In M. Zweben and M. S. Fox, editors, *Intelligent Scheduling*, chapter 8, pages 241–255. Morgan Kaufmann, 1994.